# Generalization in decision trees and DNF: Does size matter?

Mostefa Golea[1], Peter L. Bartlett[1]*, Wee Sun Lee[2] and Llew Mason[1]

[1] Department of Systems Engineering
Research School of Information
Sciences and Engineering
Australian National University
Canberra, ACT, 0200, Australia

[2] School of Electrical Engineering
University College UNSW
Australian Defence Force Academy
Canberra, ACT, 2600, Australia

## Abstract

Recent theoretical results for pattern classification with thresholded real-valued functions (such as support vector machines, sigmoid networks, and boosting) give bounds on misclassification probability that do not depend on the size of the classifier, and hence can be considerably smaller than the bounds that follow from the VC theory. In this paper, we show that these techniques can be more widely applied, by representing other boolean functions as two-layer neural networks (thresholded convex combinations of boolean functions). For example, we show that with high probability any decision tree of depth no more than $d$ that is consistent with $m$ training examples has misclassification probability no more than $O\left(\left(\frac{1}{m}\left(N_{\text{eff}}\,\text{VCdim}(\mathcal{U})\,\log^2 m \log d\right)\right)^{1/2}\right)$, where $\mathcal{U}$ is the class of node decision functions, and $N_{\text{eff}} \leq N$ can be thought of as the effective number of leaves (it becomes small as the distribution on the leaves induced by the training data gets far from uniform). This bound is qualitatively different from the VC bound and can be considerably smaller.

We use the same technique to give similar results for DNF formulae.

* Author to whom correspondence should be addressed

# 1  INTRODUCTION

Decision trees are widely used for pattern classification [2, 7]. For these problems, results from the VC theory suggest that the amount of training data should grow at least linearly with the size of the tree[4, 3]. However, empirical results suggest that this is not necessary (see [6, 10]). For example, it has been observed that the error rate is not always a monotonically increasing function of the tree size[6].

To see why the size of a tree is not always a good measure of its complexity, consider two trees, $A$ with $N_A$ leaves and $B$ with $N_B$ leaves, where $N_B \ll N_A$. Although $A$ is larger than $B$, if most of the classification in $A$ is carried out by very few leaves and the classification in $B$ is equally distributed over the leaves, intuition suggests that $A$ is actually much simpler than $B$, since tree $A$ can be approximated well by a small tree with few leaves. In this paper, we formalize this intuition.

We give misclassification probability bounds for decision trees in terms of a new complexity measure that depends on the distribution on the leaves that is induced by the training data, and can be considerably smaller than the size of the tree. These results build on recent theoretical results that give misclassification probability bounds for thresholded real-valued functions, including support vector machines, sigmoid networks, and boosting (see [1, 8, 9]), that do not depend on the size of the classifier. We extend these results to decision trees by considering a decision tree as a thresholded convex combination of the *leaf functions* (the boolean functions that specify, for a given leaf, which patterns reach that leaf). We can then apply the misclassification probability bounds for such classifiers. In fact, we derive and use a refinement of the previous bounds for convex combinations of base hypotheses, in which the base hypotheses can come from several classes of different complexity, and the VC-dimension of the base hypothesis class is replaced by the average (under the convex coefficients) of the VC-dimension of these classes. For decision trees, the bounds we obtain depend on the effective number of leaves, a data dependent quantity that reflects how uniformly the training data covers the tree's leaves. This bound is qualitatively different from the VC bound, which depends on the total number of leaves in the tree.

In the next section, we give some definitions and describe the techniques used. We present bounds on the misclassification probability of a thresholded convex combination of boolean functions from base hypothesis classes, in terms of a misclassification margin and the average VC-dimension of the base hypotheses. In Sections 3 and 4, we use this result to give error bounds for decision trees and disjunctive normal form (DNF) formulae.

# 2  GENERALIZATION ERROR IN TERMS OF MARGIN AND AVERAGE COMPLEXITY

We begin with some definitions. For a class $\mathcal{H}$ of $\{-1, 1\}$-valued functions defined on the input space $X$, the convex hull $\text{co}(\mathcal{H})$ of $\mathcal{H}$ is the set of $[-1, 1]$-valued functions of the form $\sum_i a_i h_i$, where $a_i \geq 0$, $\sum_i a_i = 1$, and $h_i \in \mathcal{H}$. A function in $\text{co}(\mathcal{H})$ is used for classification by composing it with the threshold function, $\text{sgn} : \mathbb{R} \to \{-1, 1\}$, which satisfies $\text{sgn}(\alpha) = 1$ iff $\alpha \geq 0$. So $f \in \text{co}(\mathcal{H})$ makes a mistake on the pair $(x, y) \in X \times \{-1, 1\}$ iff $\text{sgn}(f(x)) \neq y$. We assume that labelled examples $(x, y)$ are generated according to some probability distribution $\mathcal{D}$ on $X \times \{-1, 1\}$, and we let $\mathbf{P}_{\mathcal{D}}[E]$ denote the probability under $\mathcal{D}$ of an event $E$. If $S$ is a finite subset of $Z$, we let $\mathbf{P}_S[E]$ denote the empirical probability of $E$ (that is, the proportion of points in $S$ that lie in $E$). We use $\mathbf{E}_{\mathcal{D}}[\cdot]$ and $\mathbf{E}_S[\cdot]$ to denote expectation in a similar way. For a function class $H$ of $\{-1, 1\}$-valued functions defined on the input

space $X$, the growth function and VC dimension of $H$ will be denoted by $\Pi_H(m)$ and $\mathrm{VCdim}(H)$ respectively.

In [8], Schapire *et al* give the following bound on the misclassification probability of a thresholded convex combination of functions, in terms of the proportion of training data that is labelled to the correct side of the threshold by some margin. (Notice that $\mathbf{P}_\mathcal{D}\left[\mathrm{sgn}(f(x)) \neq y\right] \leq \mathbf{P}_\mathcal{D}\left[yf(x) \leq 0\right]$.)

**Theorem 1 ([8])** *Let $\mathcal{D}$ be a distribution on $X \times \{-1, 1\}$, $\mathcal{H}$ a hypothesis class with $\mathrm{VCdim}(H) = d < \infty$, and $\delta > 0$. With probability at least $1 - \delta$ over a training set $S$ of $m$ examples chosen according to $\mathcal{D}$, every function $f \in \mathrm{co}(\mathcal{H})$ and every $\theta > 0$ satisfy*

$$\mathbf{P}_\mathcal{D}\left[yf(x) \leq 0\right] \leq \mathbf{P}_S\left[yf(x) \leq \theta\right] + O\left(\frac{1}{\sqrt{m}}\left(\frac{d\log^2(m/d)}{\theta^2} + \log(1/\delta)\right)^{1/2}\right).$$

In Theorem 1, all of the base hypotheses in the convex combination $f$ are elements of a single class $\mathcal{H}$ with bounded VC-dimension. The following theorem generalizes this result to the case in which these base hypotheses may be chosen from any of $k$ classes, $\mathcal{H}_1, \ldots, \mathcal{H}_k$, which can have different VC-dimensions. It also gives a related result that shows the error decreases to twice the error estimate at a faster rate.

**Theorem 2** *Let $\mathcal{D}$ be a distribution on $X \times \{-1, 1\}$, $\mathcal{H}_1, \ldots, \mathcal{H}_k$ hypothesis classes with $\mathrm{VCdim}(H_i) = d_i$, and $\delta > 0$. With probability at least $1 - \delta$ over a training set $S$ of $m$ examples chosen according to $\mathcal{D}$, every function $f \in \mathrm{co}\left(\bigcup_{i=1}^{k} \mathcal{H}_i\right)$ and every $\theta > 0$ satisfy both*

$$\mathbf{P}_\mathcal{D}\left[yf(x) \leq 0\right] \leq \mathbf{P}_S\left[yf(x) \leq \theta\right] +$$
$$O\left(\frac{1}{\sqrt{m}}\left(\frac{1}{\theta^2}\left(d\log m + \log k\right)\log\left(m\theta^2/d\right) + \log(1/\delta)\right)^{1/2}\right),$$

$$\mathbf{P}_\mathcal{D}\left[yf(x) \leq 0\right] \leq 2\mathbf{P}_S\left[yf(x) \leq \theta\right] +$$
$$O\left(\frac{1}{m}\left(\frac{1}{\theta^2}\left(d\log m + \log k\right)\log\left(m\theta^2/d\right) + \log(1/\delta)\right)\right),$$

*where $d = \sum_i a_i d_{j_i}$ and the $a_i$ and $j_i$ are defined by $f = \sum_i a_i h_i$ and $h_i \in \mathcal{H}_{j_i}$ for $j_i \in \{1, \ldots, k\}$.*

**Proof sketch:** We shall sketch only the proof of the first inequality of the theorem. The proof closely follows the proof of Theorem 1 (see [8]). We consider a number of approximating sets of the form $\mathcal{C}_{N,l} = \left\{(1/N)\sum_{i=1}^{N} \hat{h}_i : \hat{h}_i \in \mathcal{H}_{l_i}\right\}$, where $l = (l_1, \ldots, l_N) \in \{1, \ldots, k\}^N$ and $N \in \mathbb{N}$. Define $\mathcal{C}_N = \bigcup_l \mathcal{C}_{N,l}$. For a given $f = \sum_i a_i h_i$ from $\mathrm{co}\left(\bigcup_{i=1}^{k} \mathcal{H}_i\right)$, we shall choose an approximation $g \in \mathcal{C}_N$ by choosing $\hat{h}_1, \ldots, \hat{h}_N$ independently from $\{h_1, h_2, \ldots, \}$, according to the distribution defined by the coefficients $a_i$. Let $\mathcal{Q}$ denote this distribution on $\mathcal{C}_N$. As in [8], we can take the expectation under this random choice of $g \in \mathcal{C}_N$ to show that, for any $\theta > 0$, $\mathbf{P}_\mathcal{D}\left[yf(x) \leq 0\right] \leq \mathbf{E}_{g \sim \mathcal{Q}}\left[\mathbf{P}_\mathcal{D}\left[yg(x) \leq \theta/2\right]\right] + \exp(-N\theta^2/8)$. Now, for a given $l \in \{1, \ldots, k\}^N$, the probability that there is a $g$ in $\mathcal{C}_{N,l}$ and a $\theta > 0$ for which $\mathbf{P}_\mathcal{D}\left[yg(x) \leq \theta/2\right] > \mathbf{P}_S\left[yg(x) \leq \theta/2\right] + \epsilon_{N,l}$ is at most $8(N+1)\prod_{i=1}^{N}\left(\frac{2em}{d_{l_i}}\right)^{d_{l_i}}\exp(-m\epsilon_{N,l}^2/32)$. Applying the union bound

(over the values of $l$), taking expectation over $g \sim \mathcal{Q}$, and setting $\epsilon_{N,l} = \left(\frac{32}{m} \ln \left(8(N+1) \prod_{i=1}^{N} \left(\frac{2em}{d_{l_i}}\right)^{d_{l_i}} k^N/\delta_N\right)\right)^{1/2}$ shows that, with probability at least $1 - \delta_N$, every $f$ and $\theta > 0$ satisfy $\mathbf{P}_{\mathcal{D}}[yf(x) \leq 0] \leq \mathbf{E}_g[\mathbf{P}_S[yg(x) \leq \theta/2]] + \mathbf{E}_g[\epsilon_{N,l}]$. As above, we can bound the probability inside the first expectation in terms of $\mathbf{P}_S[yf(x) \leq \theta]$. Also, Jensen's inequality implies that $\mathbf{E}_g[\epsilon_{N,l}] \leq \left(\frac{32}{m}(\ln(8(N+1)/\delta_N) + N \ln k + N \sum_i a_i d_{j_i} \ln(2em))\right)^{1/2}$. Setting $\delta_N = \delta/(N(N+1))$ and $N = \left\lceil \frac{8}{\theta^2} \ln\left(\frac{m\theta^2}{d}\right)\right\rceil$ gives the result. ∎

Theorem 2 gives misclassification probability bounds only for thresholded convex combinations of boolean functions. The key technique we use in the remainder of the paper is to find representations in this form (that is, as two-layer neural networks) of more arbitrary boolean functions. We have some freedom in choosing the convex coefficients, and this choice affects both the error estimate $\mathbf{P}_S[yf(x) \leq \theta]$ and the average VC-dimension $d$. We attempt to choose the coefficients and the margin $\theta$ so as to optimize the resulting bound on misclassification probability. In the next two sections, we use this approach to find misclassification probability bounds for decision trees and DNF formulae.

## 3   DECISION TREES

A two-class decision tree $T$ is a tree whose internal decision nodes are labeled with boolean functions from some class $\mathcal{U}$ and whose leaves are labeled with class labels $\sigma$ from $\{-1, +1\}$. For a tree with $N$ leaves, define the leaf functions, $h_i : X \to \{-1, 1\}$ by $h_i(x) = 1$ iff $x$ reaches leaf $i$, for $i = 1, \ldots, N$. Note that $h_i$ is the conjunction of all tests on the path from the root to leaf $i$.

For a sample $S$ and a tree $T$, let $P_i = \mathbf{P}_S[h_i(x) = 1]$. Clearly, $P = (P_1, \ldots, P_N)$ is a probability vector. Let $\sigma_i \in \{-1, +1\}$ denote the class assigned to leaf $i$. Define the class of leaf functions for leaves up to depth $j$ as

$$\mathcal{H}_j = \{h : \quad h = u_1 \wedge u_2 \wedge \cdots \wedge u_r \mid r \leq j, \ u_i \in \mathcal{U}\}.$$

It is easy to show that $\text{VCdim}(\mathcal{H}_j) \leq 2j \text{VCdim}(\mathcal{U}) \ln(2ej)$. Let $d_i$ denote the depth of leaf $i$, so $h_i \in \mathcal{H}_{d_i}$, and let $d = \max_i d_i$.

The boolean function implemented by a decision tree $T$ can be written as a thresholded convex combination of the form $T(x) = \text{sgn}(f(x))$, where $f(x) = \sum_{i=1}^{N} w_i \sigma_i ((h_i(x) + 1)/2) = \sum_{i=1}^{N} w_i \sigma_i h_i(x)/2 + \sum_{i=1}^{N} w_i \sigma_i/2$, with $w_i > 0$ and $\sum_{i=1}^{N} w_i = 1$. (To be precise, we need to enlarge the classes $\mathcal{H}_j$ slightly to be closed under negation. This does not affect the results by more than a constant.) We first assume that the tree is consistent with the training sample. We will show later how the results extend to the inconsistent case.

The second inequality of Theorem 2 shows that, for fixed $\delta > 0$ there is a constant $c$ such that, for any distribution $\mathcal{D}$, with probability at least $1 - \delta$ over the sample $S$ we have $\mathbf{P}_{\mathcal{D}}[T(x) \neq y] \leq 2\mathbf{P}_S[yf(x) \leq \theta] + \frac{1}{\theta^2} \sum_{i=1}^{N} w_i d_i B$, where $B = \frac{c}{m}\text{VCdim}(\mathcal{U}) \log^2 m \log d$. Different choices of the $w_i$s and the $\theta$ will yield different estimates of the error rate of $T$. We can assume (wlog) that $P_1 \geq \cdots \geq P_N$. A natural choice is $w_i = P_i$ and $P_{j+1} \leq \theta < P_j$ for some $j \in \{1, \ldots, N\}$ which gives

$$\mathbf{P}_{\mathcal{D}}[T(x) \neq y] \leq 2 \sum_{i=j+1}^{N} P_i + \frac{\bar{d}B}{\theta^2}, \tag{1}$$

where $\overline{d} = \sum_{i=1}^{N} P_i d_i$. We can optimize this expression over the choices of $j \in \{1 \dots, N\}$ and $\theta$ to give a bound on the misclassification probability of the tree.

Let $\rho(P, U) = \sum_{i=1}^{N}(P_i - 1/N)^2$ be the quadratic distance between the probability vector $P = (P_1, \dots, P_N)$ and the uniform probability vector $U = (1/N, 1/N, \dots, 1/N)$. Define $N_{\text{eff}} \equiv N(1 - \rho(P, U))$. The parameter $N_{\text{eff}}$ is a measure of the *effective number of leaves* in the tree.

**Theorem 3** *For a fixed $\delta > 0$, there is a constant $c$ that satisfies the following. Let $\mathcal{D}$ be a distribution on $X \times \{-1, 1\}$. Consider the class of decision trees of depth up to $d$, with decision functions in $\mathcal{U}$. With probability at least $1 - \delta$ over the training set $S$ (of size $m$), every decision tree $T$ that is consistent with $S$ has*

$$\mathbf{P}_{\mathcal{D}}\left[T(x) \neq y\right] \leq c \left(\frac{N_{\text{eff}} \, \text{VCdim}(\mathcal{U}) \log^2 m \log d}{m}\right)^{1/2},$$

*where $N_{\text{eff}}$ is the effective number of leaves of $T$.*

**Proof:** Supposing that $\theta \geq (\overline{d}/N)^{1/2}$ we optimize (1) by choice of $\theta$. If the chosen $\theta$ is actually smaller than $(\overline{d}/N)^{1/2}$ then we show that the optimized bound still holds by a standard VC result. If $\theta \geq (\overline{d}/N)^{1/2}$ then $\sum_{i=j+1}^{N} P_i \leq \theta^2 N_{\text{eff}}/\overline{d}$. So (1) implies that $\mathbf{P}_{\mathcal{D}}\left[T(x) \neq y\right] \leq 2\theta^2 N_{\text{eff}}/\overline{d} + \overline{d}B/\theta^2$. The optimal choice of $\theta$ is then $(\frac{1}{2}\overline{d}^2 B/N_{\text{eff}})^{1/4}$. So if $(\frac{1}{2}\overline{d}^2 B/N_{\text{eff}})^{1/4} \geq (\overline{d}/N)^{1/2}$, we have the result. Otherwise, the upper bound we need to prove satisfies $2(2N_{\text{eff}}B)^{1/2} > 2NB$, and this result is implied by standard VC results using a simple upper bound for the growth function of the class of decision trees with $N$ leaves. ∎

Thus the parameters that quantify the complexity of a tree are: a) the complexity of the test function class $\mathcal{U}$, and b) the effective number of leaves $N_{\text{eff}}$. The effective number of leaves can potentially be much smaller than the total number of leaves in the tree [5]. Since this parameter is data-dependent, the same tree can be simple for one set of $P_i$s and complex for another set of $P_i$s.

For trees that are not consistent with the training data, the procedure to estimate the error rate is similar. By defining $Q_i = \mathbf{P}_S\left[y\sigma_i = -1 \mid h_i(x) = 1\right]$ and $P'_i = P_i(1 - Q_i)/(1 - \mathbf{P}_S\left[T(x) \neq y\right])$ we obtain the following result.

**Theorem 4** *For a fixed $\delta > 0$, there is a constant $c$ that satisfies the following. Let $\mathcal{D}$ be a distribution on $X \times \{-1, 1\}$. Consider the class of decision trees of depth up to $d$, with decision functions in $\mathcal{U}$. With probability at least $1 - \delta$ over the training set $S$ (of size $m$), every decision tree $T$ has*

$$\mathbf{P}_{\mathcal{D}}\left[T(x) \neq y\right] \leq \mathbf{P}_S\left[T(x) \neq y\right] + c \left(\frac{N_{\text{eff}} \, \text{VCdim}(\mathcal{U}) \log^2 m \log d}{m}\right)^{1/3},$$

*where $c$ is a universal constant, and $N_{\text{eff}} = N(1 - \rho(P', U))$ is the effective number of leaves of $T$.*

Notice that this definition of $N_{\text{eff}}$ generalizes the definition given before Theorem 3.

## 4   DNF AS THRESHOLDED CONVEX COMBINATIONS

A DNF formula defined on $\{-1, 1\}^n$ is a disjunction of terms, where each term is a conjunction of literals and a literal is either a variable or its negation. For a given DNF formula $g$, we use $N$ to denote the number of terms in $g$, $t_i$ to represent the $i$th

term in $f$, $L_i$ to represent the set of literals in $t_i$, and $N_i$ the size of $L_i$. Each term $t_i$ can be thought of as a member of the class $\mathcal{H}_{N_i}$, the set of monomials with $N_i$ literals. Clearly, $|\mathcal{H}_j| = \binom{2n}{j}$. The DNF $g$ can be written as a thresholded convex combination of the form $g(x) = -\mathrm{sgn}(-f(x)) = -\mathrm{sgn}\left(-\sum_{i=1}^{N} w_i\left((t_i+1)/2\right)\right)$. (Recall that $\mathrm{sgn}(\alpha) = 1$ iff $\alpha \geq 0$.) Further, each term $t_i$ can be written as a thresholded convex combination of the form $t_i(x) = \mathrm{sgn}(f_i(x)) = \mathrm{sgn}\left(\sum_{l_k \in L_i} v_{ik}\left((l_k(x)-1)/2\right)\right)$. Assume for simplicity that the DNF is consistent (the results extend easily to the inconsistent case). Let $\gamma^+$ ($\gamma^-$) denote the fraction of positive (negative) examples under distribution $\mathcal{D}$. Let $\mathbf{P}_{\mathcal{D}+}[\cdot]$ ($\mathbf{P}_{\mathcal{D}-}[\cdot]$) denote probability with respect to the distribution over the positive (negative) examples, and let $\mathbf{P}_{S+}[\cdot]$ ($\mathbf{P}_{S-}[\cdot]$) be defined similarly, with respect to the sample $S$. Notice that $\mathbf{P}_{\mathcal{D}}[g(x) \neq y] = \gamma^+\mathbf{P}_{\mathcal{D}+}[g(x) = -1] + \gamma^-\mathbf{P}_{\mathcal{D}-}[(\exists i)\, t_i(x) = 1]$, so the second inequality of Theorem 2 shows that, with probability at least $1 - \delta$, for any $\theta$ and any $\theta_i$s,

$$\mathbf{P}_{\mathcal{D}}\left[g(x) \neq y\right] \leq \gamma^+\left(2\mathbf{P}_{S+}\left[f(x) \leq \theta\right] + \frac{\overline{d}B}{\theta^2}\right) + \gamma^-\sum_{i=1}^{N}\left(2\mathbf{P}_{S-}\left[-f_i(x) \leq \theta_i\right] + \frac{B}{\theta_i^2}\right)$$

where $\overline{d} = \sum_{i=1}^{N} w_i N_i$ and $B = c\left(\log n \log^2 m + \log(N/\delta)\right)/m$. As in the case of decision trees, different choices of $\theta$, the $\theta_i$s, and the weights yield different estimates of the error. For an arbitrary order of the terms, let $P_i$ be the fraction of positive examples covered by term $t_i$ but not by terms $t_{i-1}, \ldots, t_1$. We order the terms such that for each $i$, with $t_{i-1}, \ldots, t_1$ fixed, $P_i$ is maximized, so that $P_1 \geq \cdots \geq P_N$, and we choose $w_i = P_i$. Likewise, for a given term $t_i$ with literals $l_1, \ldots, l_{N_i}$ in an arbitrary order, let $P_k^{(i)}$ be the fraction of negative examples uncovered by literal $l_k$ but not uncovered by $l_{k-1}, \ldots, l_1$. We order the literals of term $t_i$ in the same greedy way as above so that $P_1^{(i)} \geq \cdots \geq P_{N_i}^{(i)}$, and we choose $v_{ik} = P_k^{(i)}$. For $P_{j+1} \leq \theta < P_j$ and $P_{j_i+1}^{(i)} \leq \theta_i < P_{j_i+1}^{(i)}$, where $1 \leq j \leq N$ and $1 \leq j_i \leq N_i$, we get

$$\mathbf{P}_{\mathcal{D}}\left[g(x) \neq y\right] \leq \gamma^+\left(2\sum_{i=j+1}^{N} P_i + \frac{\overline{d}B}{\theta^2}\right) + \gamma^-\sum_{i=1}^{N}\left(2\sum_{k=j_i+1}^{N_i} P_k^{(i)} + \frac{B}{\theta_i^2}\right)$$

Now, let $P = (P_1, \ldots, P_N)$ and for each term $i$ let $P^{(i)} = (P_1^{(i)}, \ldots, P_{N_i}^{(i)})$. Define $N_{\mathrm{eff}} = N(1 - \rho(P, U))$ and $N_{\mathrm{eff}}^{(i)} = N_i(1 - \rho(P^{(i)}, U))$, where $U$ is the relevant uniform distribution in each case. The parameter $N_{\mathrm{eff}}$ is a measure of the effective number of terms in the DNF formula. It can be much smaller than $N$; this would be the case if few terms cover a large fraction of the positive examples. The parameter $N_{\mathrm{eff}}^{(i)}$ is a measure of the effective number of literals in term $t_i$. Again, it can be much smaller than the actual number of literals in $t_i$: this would be the case if few literals of the term uncover a large fraction of the negative examples.

Optimizing over $\theta$ and the $\theta_i$s as in the proof of Theorem 3 gives the following result.

**Theorem 5** *For a fixed $\delta > 0$, there is a constant $c$ that satisfies the following. Let $\mathcal{D}$ be a distribution on $X \times \{-1, 1\}$. Consider the class of DNF formulae with up to $N$ terms. With probability at least $1 - \delta$ over the training set $S$ (of size $m$), every DNF formulae $g$ that is consistent with $S$ has*

$$\mathbf{P}_{\mathcal{D}}\left[g(x) \neq y\right] \leq \gamma^+(N_{\mathrm{eff}}dB)^{1/2} + \gamma^-\sum_{i=1}^{N}(N_{\mathrm{eff}}^{(i)}B)^{1/2}$$

*where $d = \max_{i=1}^{N} N_i$, $\gamma^\pm = \mathbf{P}_{\mathcal{D}}[y = \pm 1]$ and $B = c(\log n \log^2 m + \log(N/\delta))/m$.*

## 5 CONCLUSIONS

The results in this paper show that structural complexity measures (such as size) of decision trees and DNF formulae are not always the most appropriate in determining their generalization behaviour, and that measures of complexity that depend on the training data may give a more accurate description. Our analysis can be extended to multi-class classification problems. A similar analysis implies similar bounds on misclassification probability for decision lists, and it seems likely that these techniques will also be applicable to other pattern classification methods.

The complexity parameter, $N_{\text{eff}}$ described here does not always give the best possible error bounds. For example, the effective number of leaves $N_{\text{eff}}$ in a decision tree can be thought of as a single number that summarizes the probability distribution over the leaves induced by the training data. It seems unlikely that such a number will give optimal bounds for all distributions. In those cases, better bounds could be obtained by using numerical techniques to optimize over the choice of $\theta$ and $w_i$s. It would be interesting to see how the bounds we obtain and those given by numerical techniques reflect the generalization performance of classifiers used in practice.

### Acknowledgements

Thanks to Yoav Freund and Rob Schapire for helpful comments.

### References

[1] P. L. Bartlett. For valid generalization, the size of the weights is more important than the size of the network. In *Neural Information Processing Systems 9*, pages 134–140. Morgan Kaufmann, San Mateo, CA, 1997.

[2] L. Breiman, J.H. Friedman, R.A. Olshen, and C.J. Stone. *Classification and Regression Trees*. Wadsworth, Belmont, 1984.

[3] A. Ehrenfeucht and D. Haussler. Learning decision trees from random examples. *Information and Computation*, 82:231–246, 1989.

[4] U.M. Fayyad and K.B. Irani. What should be minimized in a decision tree? In *AAAI-90*, pages 249–754, 1990.

[5] R. C. Holte. Very simple rules perform well on most commonly used databases. *Machine learning*, 11:63–91, 1993.

[6] P.M. Murphy and M.J. Pazzani. Exploring the decision forest: An empirical investigation of Occam's razor in decision tree induction. *Journal of Artificial Intelligence Research*, 1:257–275, 1994.

[7] J.R. Quinlan. *C4.5: Programs for Machine Learning*. Morgan Kaufmann, 1992.

[8] R. E. Schapire, Y. Freund, P. L. Bartlett, and W. S. Lee. Boosting the margin: a new explanation for the effectiveness of voting methods. In *Machine Learning: Proceedings of the Fourteenth International Conference*, pages 322–330, 1997.

[9] J. Shawe-Taylor, P. L. Bartlett, R. C. Williamson, and M. Anthony. A framework for structural risk minimisation. In *Proc. 9th COLT*, pages 68–76. ACM Press, New York, NY, 1996.

[10] G.L. Webb. Further experimental evidence against the utility of Occam's razor. *Journal of Artificial Intelligence Research*, 4:397–417, 1996.
